# Ideal Observers for Detecting Motion: Correspondence Noise

**Hongjing Lu**
Department of Psychology, UCLA
Los Angeles, CA 90095
hongjing@psych.ucla.edu

**Alan Yuille**
Department of Statistics, UCLA
Los Angeles, CA 90095
yuille@stat.ucla.edu

## Abstract

We derive a Bayesian Ideal Observer (BIO) for detecting motion and solving the correspondence problem. We obtain Barlow and Tripathy's classic model as an approximation. Our psychophysical experiments show that the trends of human performance are similar to the Bayesian Ideal, but overall human performance is far worse. We investigate ways to degrade the Bayesian Ideal but show that even extreme degradations do not approach human performance. Instead we propose that humans perform motion tasks using generic, general purpose, models of motion. We perform more psychophysical experiments which are consistent with humans using a Slow-and-Smooth model and which rule out an alternative model using Slowness.

## 1 Introduction

Ideal Observers give fundamental limits for performing visual tasks (somewhat similar to Shannon's limits on information transfer). They give benchmarks against which to evaluate human performance. This enables us to determine objectively what visual tasks humans are good at, and may help point the way to underlying neuronal mechanisms. For a recent review, see [1].

In an influential paper, Barlow and Tripathy [2] tested the ability of human subjects to detect dots moving coherently in a background of random dots. They derived an "ideal observer" model using techniques from Signal Detection theory [3]. They showed that their model predicted the trends of the human performance as properties of the stimuli changed, but that humans performed far worse than their model. They argued that degrading their model, by lowering the spatial resolution, would give predictions closer to human performance. Barlow and Tripathy's model has generated considerable interest, see [4,5,6,7].

We formulate this motion problem in terms of Bayesian Decision Theory and derive a Bayesian Ideal Observer (BIO) model. We describe why Barlow and Tripathy's (BT) model is not fully ideal, show that it can be obtained as an approximation to the BIO, and determine conditions under which it is a good approximation. We perform psychophysical experiments under a range of conditions and show that the trends of human subjects are more similar to those of the BIO. We investigate whether degrading the Bayesian Ideal enables us to reach human performance, and conclude that it does not (without implausibly large

deformations). We comment that Barlow and Tripathy's degradation model is implausible due to the nature of the approximations used.

Instead we show that a generic motion detection model which uses a slow-and-smooth assumption about the motion field [8,9] gives similar performance to human subjects under a range of experimental conditions. A simpler approach using a slowness assumption alone does not match new experimental data that we present. We conclude that human observers are not ideal, in the sense that they do not perform inference using the model that the experimenter has chosen to generate the data, but may instead use a general purpose model perhaps adapted to the motion statistics of natural images.

## 2   Bayes Decision Theory and Ideal Observers

We now give the basic elements of Bayes Decision Theory. The input data is $D$ and we seek to estimate a binary state $W$ (e.g. coherent or incoherent motion, horizontal motion to right or to left). We assume models $P(D|W)$ and $P(W)$. We define a decision rule $\alpha(D)$ and a loss function $L(\alpha(I), W) = 1 - \delta_{\alpha(D),W}$. The risk is $R(\alpha) = \sum_{D,W} L(\alpha(D), W)P(D|W)P(W)$.

Optimal performance is given by the Bayes rule: $\alpha^* = \arg\min R(\alpha)$. The fundamental limits are given by Bayes Risk: $R^* = R(\alpha^*)$. Bayes risk is the best performance that can be achieved. It corresponds to ideal performance.

Barlow and Tripathy's (BT) model does not achieve Bayes risk. This is because they used simplification to derive it using concepts from Signal Detection theory (SDT). SDT is essentially the application of Bayes Decision Theory to the task of signal detection but, for historical reasons, SDT restricts itself to a limited class of probability models and is unable to capture the complexity of the motion problem.

## 3   Experimental Setup and Correspondence Noise

We now give the details of Barlow and Tripathy's stimuli, their model, and their experiments. The stimuli consist of two image frames with $N$ dots in each frame. The dots in the first frame are at random positions. For coherent stimuli, see figure (1), a proportion $CN$ of dots move coherently left or right horizontally with a fixed translation motion with displacement $T$. The remaining $N(1 - C)$ dots in the second frame are generated at random. For incoherent stimuli, the dots in both frames are generated at random.

Estimating motion for these stimuli requires solving the correspondence problem to match dots between frames. For coherent motion, the noise dots act as *correspondence noise* and make the matching harder, see the rightmost panel in figure (1).

Barlow and Tripathy perform two types of binary forced choice experiments. In *detection experiments*, the task is to determine whether the stimuli is coherent or incoherent motion. For *discrimination experiments*, the goal is to determine if the motion is to the right or the left.

The experiments are performed by adjusting the fraction $C$ of coherently moving dots until the human subject's performance is at threshold (i.e. 75 percent correct). Barlow and Tripathy's (BT) model gives the proportion of dots at threshold to be $C_\theta = 1/\sqrt{Q - N}$ where $Q$ is the size of the image lattice. This is approximately $1/\sqrt{Q}$ (because $N << Q$) and so is independent of the density of dots. Barlow and Tripathy compare the thresholds of the human subjects with those of their model for a range of experimental conditions which we will discuss in later sections.

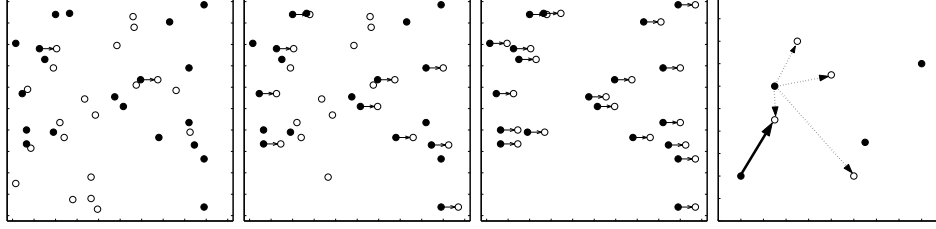

Figure 1: The left three panels show coherent stimuli with $N = 20, C = 0.1$, $N = 20, C = 0.5$ and $N = 20, C = 1.0$ respectively. The closed and open circles denote dots in the first and second frame respectively. The arrows show the motion of those dots which are moving coherently. Correspondence noise is illustrated by the far right panel showing that a dot in the first frame has many candidate matches in the second frame.

## 4  The Bayesian Ideal Model

We now compute the Bayes rule and Bayes risk by taking into account exactly how the data is generated. We denote the dot positions in the first and second frame by $D = \{x_i : i = 1, ..., N\}, \{y_a : a = 1, ..., N\}$. We define correspondence variables $V_{ia} : V_{ia} = 1$ if $x_i \to y_a$, $V_{ia} = 0$ otherwise.

The generative model for the data is given by:

$$P(D|\text{Coh}, T) = \sum_{V_{ia}} P(\{y_a\}|\{x_i\}, \{V_{ia}\}, T)P(\{V_{ia}\})P(\{x_i\}) \text{ coherent,}$$
$$P(D|\text{Incoh}) = P(\{y_a\})P(\{x_i\}), \text{ incoherent.} \tag{1}$$

The prior distributions for the dot positions $P(\{x_i\}), P(\{y_a\})$ allow all configurations of the dots to be equally likely. They are therefore of form $P(\{x_i\}) = P(\{y_a\}) = \frac{(Q-N)!}{Q!}$ where $Q$ is the number of lattice points. The model $P(\{y_a\}|\{x_i\}, \{V_{ia}\}, T)$ for coherent motion is $P(\{y_a\}|\{x_i\}, \{V_{ia}\}, T) = \frac{(Q-N)!}{(Q-CN)!} \prod_{ia} (\delta_{y_a, x_i+T})^{V_{ia}}$. We set the priors $P(\{V_{ia}\})$ to be the uniform distribution. There is a constraint $\sum_{ia} V_{ia} = CN$ (since only $CN$ dots move coherently).

This gives:

$$P(D|\text{Incoh}) = \frac{(Q-N)!}{Q!} \frac{(Q-N)!}{Q!},$$
$$P(D|\text{Coh}, T) = \{\frac{(N-CN)!}{(N)!} \frac{(N-CN)!}{(N)!}\}^2 (CN)! \sum_{V_{ia}} \prod_{ia} (\delta_{y_a+T, x_i})^{V_{ia}}.$$

These can be simplified further by observing that $\sum_{V_{ia}} \prod_{ia} (\delta_{y_a, x_i+T})^{V_{ia}} = \frac{\Psi!}{(\Psi-M)!M!}$, where $\Psi$ is the total number of matches – i.e. the number of dots in the first frame that have a corresponding dot at displacement $T$ in the second frame (this includes "fake" matches due to change alignment of noise dots in the two frames).

The Bayes rule for performing the tasks are given by testing the log-likelihood ratios: (i) $\log \frac{P(D|\text{Incoh})}{P(D|\text{Coh}, T)}$ for detection (i.e. coherent versus incoherent), and (ii) $\log \frac{P(D|\text{Coh}, -T)}{P(D|\text{Coh}, T)}$ for discrimination (i.e. motion to right or to left). For detection, the log-likelihood ratio is a function of $\Psi$. For discrimination, the log-likelihood ratio is a function of the number of matches to the right $\Psi_r$ and to the left $\Psi_l$. It is straightforward to calculate the Bayes risk and determine coherence thresholds.

We can rederive Barlow and Tripathy's model as an approximation to the Bayesian Ideal. They make two approximations: (i) they model the distribution of $\psi$ as Binomial, (ii) they use $d'$. Both approximations are very good near threshold, except for small $N$. The use of $d'$ can be justified if $P(\Psi|\text{Coh}, T)$ and $P(\Psi|\text{Incoh})$ are Gaussians with similar variance. This is true for large $N = 1000$ and a range of $C$ but not so good for small $N = 100$, see figure (2).

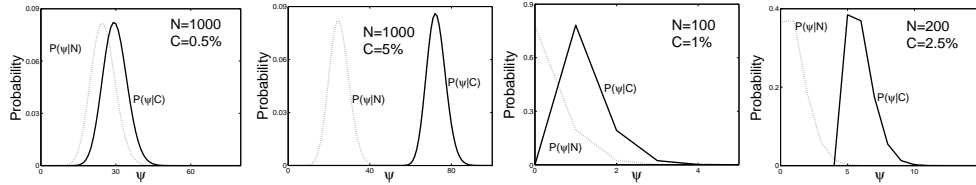

Figure 2: We plot $P(\Psi|\text{Coh}, T)$ and $P(\Psi|\text{Incoh})$, shown as $P(\Psi|C)$ and $P(\Psi|N)$ respectively, for a range of $N$ and $C$. One of Barlow and Tripathy's two approximations are justified if the distributions are Gaussian with the same variance. This is true for large $N$ (left two panels) but fails for small $N$ (right two panels). Note that human thresholds are roughly 30 times higher than for BIO (the scales on graphs differ).

We computed the coherence threshold for the BIO and the BT models for $N = 100$ to $N = 1000$, see the second and fourth panels in figure (3). As described earlier, the BT threshold is approximately independent of the number $N$ of dots. Our computations showed that the BIO threshold is also roughly constant except for small $N$ (this is not surprising in light of figure (2). This motivated psychophysics experiments to determine how humans performed for small $N$ (this range of dots was not explored in Barlow and Tripathy's experiments).

All our data points are from 300 trials using QUEST, so errors bars are so small that we do not include them.

We performed the detection and discrimination tasks with translation motion $T = 16$ (as in Barlow and Tripathy). For detection and discrimation, the human subject's thresholds showed similar trends to the thresholds for BIO and BT. But human performance at small $N$ are more consistent with BIO, see figure (3).

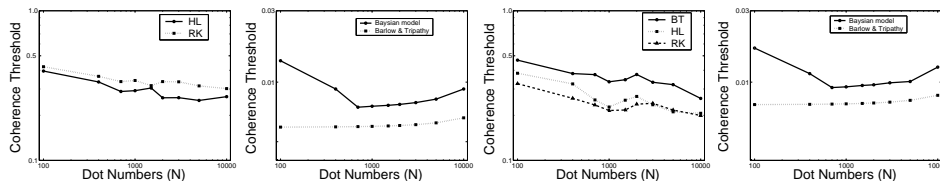

Figure 3: The left two panels show detection thresholds – human subjects (far left) and BIO and BT thresholds (left). The right two panels show discrimination thresholds – human subjects (right) and BIO and BT (far right).

But probably the most striking aspect of figure (3) is how poorly humans perform compared to the models. The thresholds for BIO are always higher than those for BT, but these differences are almost negligible compared to the differences with the human subjects. The experiments also show that the human subject trends differ from the models at large $N$. But these are extreme conditions where there are dots on most points on the image lattice.

## 5 Degradating the Ideal Observer Models

We now degrade the Bayes Ideal model to see if we can obtain human performance. We consider two mechanisms: (A) Humans do not know the precise value of the motion translation $T$. (B) Humans have poor spatial uncertainty. We will also combine both mechanisms.

For (A), we model lack of knowledge of the velocity $T$ by summing over different motions. We generate the stimuli as before from $P(D|\text{Incoh})$ or $P(D|\text{Coh}, T)$, but we make the decision by thresholding: $\log \dfrac{\sum_T P(D|\text{Coh},T)P(T)}{P(D|\text{Incoh})}$.

For (B), we model lack of spatial resolution by replacing $P(\{y_a\}|\{x_i\}, \{V_{ia}\}, T) = \frac{(Q-N)!}{(Q-CN)!} \prod_{ia} V_{ia}\delta_{y_a,x_i+t}$ by $P(\{y_a\}|\{x_i\}, \{V_{ia}\}, T) = \frac{(Q-N)!}{(Q-CN)!} \prod_{ia} V_{ia}f_W(y_a, x_i+t)$. Here $W$ is the width of a spatial window, so that $f_W(a,b) = 1/W^2$, if $|a-b| < W$; $f_W(a,b) = 0$, otherwise.

Our calculations, see figure (4), show that neither (A) nor (B) not their combination are sufficient to account for the poor performance of human subjects. Lack of knowledge of the correct motion (and consequently summing over several models) does little to degrade performance. Decreasing spatial resolution does degrade performance but even huge degradations are insufficient to reach human levels. Barlow and Tripathy [2] argue that they can degrade their model to reach human performance but the degradations are huge and they occur in conditions (e.g. $N = 50$ or $N = 100$) where their model is not a good approximation to the true Bayesian Ideal Observer.

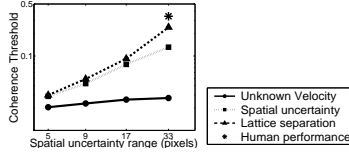

Figure 4: Comparing the degraded models to human performance. We use a log-log plot because the differences between humans and model thresholds is very large.

## 6 Slowness and Slow-and-Smooth

We now consider an alternative explanation for why human performance differs so greatly from the Bayesian Ideal Observer. Perhaps human subjects do not use the ideal model (which is only known to the designer of the experiments) and instead use a general purpose motion model. We now consider two possible models: (i) a slowness model, and (ii) a slow and smooth model.

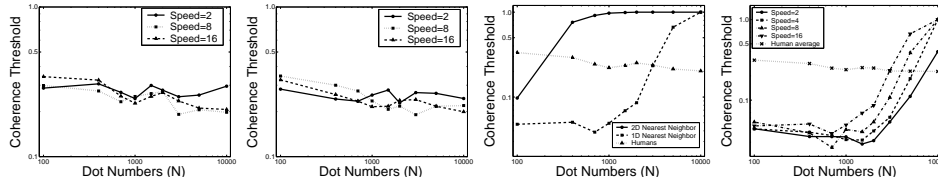

Figure 5: The coherence threshold as a function of $N$ for different translation motions $T$. From left to right, human subject (HL), human subject (RK), 2DNN (shown for $T = 16$ only), and 1DNN. In the two right panels we have drawn the average human performance for comparision.

The slowness model is partly motivated by Ullman's minimal mapping theory [10] and partly by the design of practical computer vision tracking systems. This model solves the correspondence problem by simply matching a dot in the first frame to the closest dot in the second frame. We consider a 2D nearest neighbour model (2DNN) and a 1D nearest neighbour model (1DNN), for which the matching is constrained to be in horizontal directions only. After the motion has been calculated we perform a log-likelihood test to solve the discrimination and detection tasks. This enables us to calculate coherence thresholds, see figure (5). Both 1DNN and 2DNN predict that correspondence will be easy for small translation motions even when the number of dots is very large. This motivates a new class of experiments where we vary the translation motion.

Our experiments show that 1DNN and 2DNN are poor fits to human performance. Human performance thresholds are relatively insensitive to the number $N$ of dots and the translation motion $T$, see the two left panels in figure (5). By contrast, the 1DNN and 2DNN thresholds are either far lower than humans for small $N$ or far higher at large $N$ with a transition that depends on $T$. We conclude that the 1DNN and 2DNN models do not match human performance.

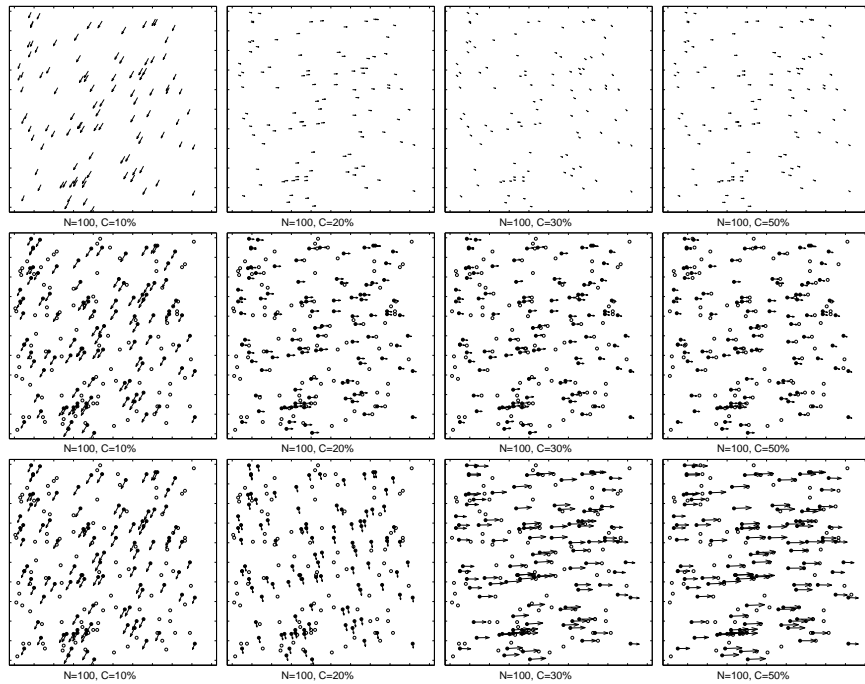

Figure 6: The motion flows from Slow-and-Smooth for $N = 100$ as functions of $C$ and $T$. From left to right, $C = 0.1, C = 0.2, C = 0.3, C = 0.5$. From top to bottom, $T = 4, T = 8, T = 16$. The closed and open circles denote dots in the first and second frame respectively. The arrows indicate the motion flow specified by the Slow-and-Smooth model.

We now consider the Slow-and-Smooth model [8,9] which has been shown to account for a range of motion phenomena. We use a formulation [8] that was specifically designed for dealing with the correspondence problem.

This gives a model of form $P(V, v|\{x_i\}, \{y_a\}) = (1/Z)e^{-E[V,v]/T_m}$, where

$$E[V, v] = \sum_{i=1}^{N} \sum_{a=1}^{N} V_{ia}(y_a - x_i - v(x_i))^2 + \lambda||Lv||^2 + \zeta \sum_{i=1}^{N} V_{i0}, \quad (2)$$

$L$ is an operator that penalizes slow-and-smooth motion and depends on a paramters $\sigma$, see Yuille and Grzywacz for details [8]. We impose the constraint that $\sum_{i=a}^{N} V_{ia} = 1$, $\forall i$, which enforces that each point $i$ in the first frame is either unmatched, if $V_{i0} = 1$, or is matched to a point $a$ in the second frame.

We implemented this model using an EM algorithm to estimate the motion field $v(x)$ that maximizes $P(v|\{x_i\}, \{y_a\}) = \sum_V P(V, v|\{x_i\}, \{y_a\})$. The parameter settings are $T_m = 0.001$, $\lambda = 0.5$, $\zeta = 0.01$, $\sigma = 0.2236$. (The size of the units of length are normalized by the size of the image). The size of $\sigma$ determines the spatial scale of the interaction between dots [8]. This parameter settings estimate correct motion directions in the condition that all dots move coherently, $C = 1.0$.

The following results, see figure (6), show that for 100 dots ($N = 100$) the results of the slow-and-smooth model are similar to those of the human subjects for a range of different translation motions. Slow-and-Smooth starts giving coherence thresholds between $C = 0.2$ and $C = 0.3$ consistent with human performance. Lower thresholds occurred for slower coherent translations in agreement with human performance.

Slow-and-Smooth also gives thresholds similar to human performance when we alter the number $N$ of dots, see figure (7). Once again, Slow-and-Smooth starts giving the correct horizontal motion between $c = 0.2$ and $c = 0.3$.

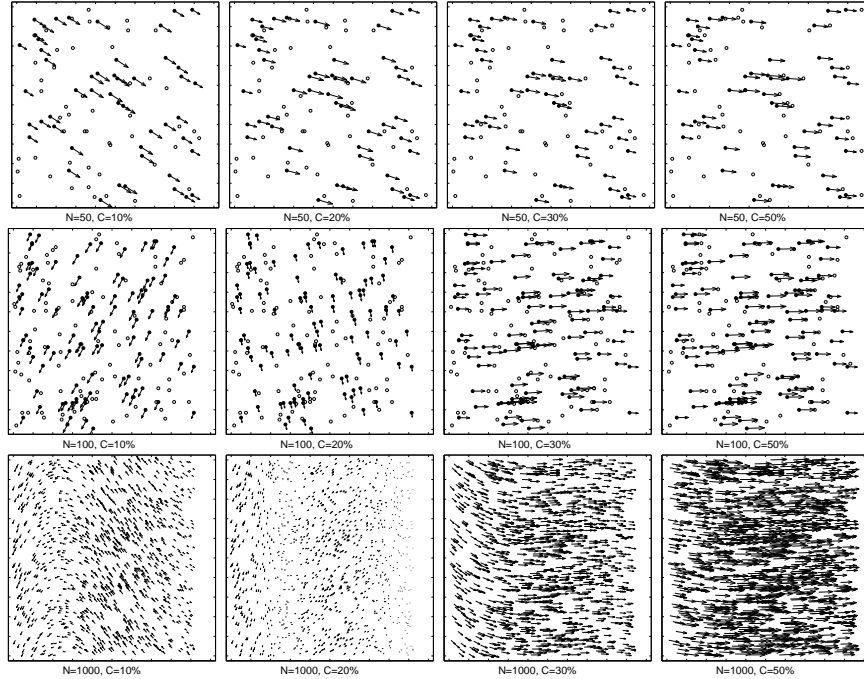

Figure 7: The motion fields of Slow-and-Smooth for $T = 16$ as a function of $c$ and $N$. From left to right, $C = 0.1, C = 0.2, C = 0.3, C = 0.5$. From top to bottom, $N = 50, N = 100, N = 1000$. Same conventions as for previous figure.

# 7   Summary

We defined a Bayes Ideal Observer (BIO) for correspondence noise and showed that Barlow and Tripathy's (BT) model [2] can be obtained as an approximation. We performed psychophysical experiments which showed that the trends of human performance were more similar to those of BIO (when it differed from BT). We attempted to account for human's poor performance (compared to BIO) by allowing for degradations of the model such as poor spatial resolution and uncertainty about the precise translation velocity. We concluded that these degradation had to be implausibly large to account for the poorness of human performance. We noted that Barlow and Tripathy's degradation model [2] takes them into a regime where their model is a bad approximation to the BIO. Instead, we investigated the possibility that human observers perform these motion tasks using generic probability models for motion possibly adapted to the statistics of motion in the natural world. Further psychophysical experiments showed that human performance was inconsistent with a model than prefers slow motion. But human performance was consistent with the Slow-and-Smooth model [8,9].

We conclude with two metapoints. Firstly, it is possible to design ideal observer models for complex stimuli using techniques from Bayes decision theory. There is no need to restrict oneself to the traditional models described in classic signal detection books such as Green and Swets [3]. Secondly, human performance at visual tasks may be based on *generic models*, such as Slow-and-Smooth, rather than the ideal models for the experimental tasks (known only to the experimenter).

## Acknowledgements

We thank Zili Liu for helpful discussions. We gratefully acknowledge funding support from the American Association of University Women (HL), NSF0413214 and W.M. Keck Foundation (ALY).

## References

[1] Geisler, W.S. (2002) "Ideal Observer Analysis". In L. Chalupa and J. Werner (Eds). The Visual Neurosciences. Boston. MIT Press. 825-837.

[2] Barlow, H., and Tripathy, S.P. (1997) Correspondence noise and signal pooling in the detection of coherent visual motion. Journal of Neuroscience, 17(20), 7954-7966.

[3] Green, D.M., and Swets, J.A. (1966) Signal detection theory and psychophysics. New York: Wiley.

[4] Morrone, M.C., Burr, D. C., and Vaina, L. M. (1995) Two stages of visual processing for radial and circular motion. Nature, 376(6540), 507-509.

[5] Neri, P., Morrone, M.C., and Burr, D.C. (1998) Seeing biological motion. Nature, 395(6705), 894-896.

[6] Song, Y., and Perona, P. (2000) A computational model for motion detection and direction discrimination in humans. IEEE computer society workshop on Human Motion, Austin, Texas.

[7] Wallace, J.M and Mamassian, P. (2004) The efficiency of depth discrimination for non-transparent and transparent stereoscopic surfaces. Vision Research, 44, 2253-2267.

[8] Yuille, A.L. and Grzywacz, N.M. (1988) A computational theory for the perception of coherent visual motion. Nature, 333,71-74,

[9] Weiss, Y., and Adelson, E.H. (1998) Slow and smooth: A Bayesian theory for the combination of local motion signals in human vision Technical Report 1624. Massachusetts Institute of Technology.

[10] Ullman, S. (1979) The interpretation of Visual Motion. MIT Press, Cambridge, MA, 1979.
